# Fast Parameter Estimation Using Green's Functions

**K. Y. Michael Wong**
Department of Physics
Hong Kong University
of Science and Technology
Clear Water Bay, Hong Kong
*phkywong@ust.hk*

**Fuli Li**
Department of Applied Physics
Xian Jiaotong University
Xian, China 710049
*flli@xjtu.edu.cn*

## Abstract

We propose a method for the fast estimation of hyperparameters in large networks, based on the linear response relation in the cavity method, and an empirical measurement of the Green's function. Simulation results show that it is efficient and precise, when compared with cross-validation and other techniques which require matrix inversion.

## 1 Introduction

It is well known that correct choices of hyperparameters in classification and regression tasks can optimize the complexity of the data model, and hence achieve the best generalization [1]. In recent years various methods have been proposed to estimate the optimal hyperparameters in different contexts, such as neural networks [2], support vector machines [3, 4, 5] and Gaussian processes [5]. Most of these methods are inspired by the technique of cross-validation or its variant, leave-one-out validation. While the leave-one-out procedure gives an almost unbiased estimate of the generalization error, it is nevertheless very tedious. Many of the mentioned attempts aimed at approximating this tedious procedure without really having to sweat through it. They often rely on theoretical bounds, inverses to large matrices, or iterative optimizations.

In this paper, we propose a new approach to hyperparameter estimation in large systems. It is known that large networks are mean-field systems, so that when one example is removed by the leave-one-out procedure, the background adjustment can be analyzed by a self-consistent perturbation approach. Similar techniques have been applied to the neural network [6], Bayesian learning [7] and the support vector machine [5]. They usually involve a macroscopic number of unknown variables, whose solution is obtained through the inversion of a matrix of macroscopic size, or iteration. Here we take a further step to replace it by a direct measurement of the Green's function via a small number of learning processes. The proposed procedure is fast since it does not require repetitive cross-validations, matrix inversions, nor iterative optimizations for each set of hyperparaemters. We will also present simulation results which show that it is an excellent approximation.

The proposed technique is based on the cavity method, which was adapted from disordered systems in many-body physics. The basis of the cavity method is a self-consistent argument addressing the situation of removing an example from the system. The change on removing an example is described by the Green's function, which is an extremely general technique used in a wide range of quantum and classical problems in many-body physics [8]. This provides an excellent framework for the leave-one-out procedure. In this paper, we consider two applications of the cavity method to hyperparameter estimation, namely the optimal weight decay and the optimal learning time in feedforward networks. In the latter application, the cavity method provides, as far as we are aware of, the only estimate of the hyperparameter beyond empirical stopping criteria and brute force cross-validation.

## 2 Steady-State Hyperparameter Estimation

Consider the network with adjustable parameters $\vec{w}$. An energy function $E$ is defined with respect to a set of $p$ examples with inputs and outputs respectively given by $\vec{\xi}^\mu$ and $y^\mu$, $\mu = 1, \cdots, p$, where $\vec{\xi}^\mu$ is an $N$-dimensional input vector with components $\xi_j^\mu$, $j = 1, \cdots, N$, and $N \gg 1$ is macroscopic. We will first focus on the dynamics of a single-layer feedforward network and generalize the results to multilayer networks later. In single-layer networks, $E$ has the form

$$E = \sum_\mu \epsilon(x^\mu, y^\mu) + R(\vec{w}).$$ (1)

Here $\epsilon(x^\mu, y^\mu)$ represents the error function with respect to example $\mu$. It is expressed in terms of the activation $x^\mu \equiv \vec{w} \cdot \vec{\xi}^\mu$. $R(\vec{w})$ represents a regularization term which is introduced to limit the complexity of the network and hence enhance the generalization ability. Learning is achieved by the gradient descent dynamics

$$\frac{dw_j(t)}{dt} = -\frac{1}{N} \frac{\partial E}{\partial w_j}.$$ (2)

The time-dependent Green's function $G_{jk}(t,s)$ is defined as the response of the weight $w_j$ at time $t$ due to a unit stimulus added at time $s$ to the gradient term with respect to weight $w_k$, in the limit of a vanishing magnitude of the stimulus. Hence if we compare the evolution of $w_j(t)$ with another system $\tilde{w}_j(t)$ with a continuous perturbative stimulus $\delta h_j(t)$, we would have

$$\frac{d\tilde{w}_j(t)}{dt} = -\frac{1}{N} \frac{\partial E}{\partial \tilde{w}_j} + \delta h_j(t),$$ (3)

and the linear response relation

$$\tilde{w}_j(t) = w_j(t) + \sum_k \int ds\, G_{jk}(t,s) \delta h_k(s).$$ (4)

Now we consider the evolution of the network $w_j^{\backslash \mu}(t)$ in which example $\mu$ is omitted from the training set. For a system learning macroscopic number of examples, the changes induced by the omission of an example are perturbative, and we can assume that the system has a linear response. Compared with the original network $w_j(t)$, the gradient of the error of example $\mu$ now plays the role of the stimulus in (3). Hence we have

$$w_j^{\backslash \mu}(t) = w_j(t) + \frac{1}{N} \sum_k \int ds\, G_{jk}(t,s) \xi_k^\mu \frac{\partial \epsilon(x^\mu(s), y^\mu)}{\partial x^\mu(s)}.$$ (5)

Multiplying both sides by $\xi_j^\mu$ and summing over $j$, we obtain

$$h^\mu(t) = x^\mu(t) + \int ds \left[ \frac{1}{N} \sum_{jk} \xi_j^\mu G_{jk}(t,s) \xi_k^\mu \right] \frac{\partial \epsilon(x^\mu(s), y^\mu)}{\partial x^\mu(s)}. \qquad (6)$$

Here $h^\mu(t) \equiv \vec{w}^{\backslash \mu}(t) \cdot \vec{\xi}^\mu$ is called the *cavity activation* of example $\mu$. When the dynamics has reached the steady state, we arrive at

$$h^\mu = x^\mu + \gamma \frac{\partial \epsilon(x^\mu, y^\mu)}{\partial x^\mu}, \qquad (7)$$

where $\gamma = \lim_{t \to \infty} \int ds [\sum_{jk} \xi_j^\mu G_{jk}(t,s) \xi_k^\mu]/N$ is the susceptibility.

At time $t$, the generalization error is defined as the error function averaged over the distribution of input $\vec{\xi}$, and their corresponding output $y$, i.e.,

$$\epsilon_g(t) = \int d\vec{\xi} P(\vec{\xi}) \int dy P(y|\xi) \epsilon(x, y), \qquad (8)$$

where $x \equiv \vec{w} \cdot \vec{\xi}$ is the network activation. The leave-one-out generalization error is an estimate of $\epsilon_g$ given in terms of the cavity activations $h^\mu$ by $\hat{\epsilon}_g = \sum_\mu \epsilon(h^\mu, y^\mu)/p$. Hence if we can estimate the Green's function, the cavity activation in (7) provides a convenient way to estimate the leave-one-out generalization error without really having to undergo the validation process.

While self-consistent equations for the Green's function have been derived using diagrammatic methods [9], their solutions cannot be *computed* except for the specific case of time-translational invariant Green's functions, such as those in Adaline learning or linear regression. However, the linear response relation (4) provides a convenient way to *measure* the Green's function in the general case. The basic idea is to perform two learning processes in parallel, one following the original process (2) and the other having a constant stimulus as in (3) with $\delta h_j(t) = \eta \delta_{jk}$, where $\delta_{jk}$ is the Kronecka delta. When the dynamics has reached the steady state, the measurement $\tilde{w}_j - w_j$ yields the quantity $\eta \sum_k \int ds G_{jk}(t,s)$.

A simple averaging procedure, replacing all the pairwise measurements between the stimulation node $k$ and observation node $j$, can be applied in the limit of large $N$. We first consider the case in which the inputs are independent and normalized, i.e., $\langle \xi_j \rangle = 0$, $\langle \xi_j \xi_k \rangle = \delta_{jk}$. In this case, it has been shown that the off-diagonal Green's functions can be neglected, and the diagonal Green's functions become self-averaging, i.e., $G_{jk}(t,s) = G(t,s)\delta_{jk}$, independent of the node labels [9], rendering $\gamma = \lim_{t \to \infty} \int ds G(t,s)$.

In the case that the inputs are correlated and not normalized, we can apply standard procedures of whitening transformation to make them independent and normalized [1]. In large networks, one can use the diagrammatic analysis in [9] to show that the (unknown) distribution of inputs does not change the self-averaging property of the Green's functions after the whitening transformation. Thereafter, the measurement of Green's functions proceeds as described in the simpler case of independent and normalized inputs. Since hyperparameter estimation usually involves a series of computing $\hat{\epsilon}_g$ at various hyperparameters, the one-time preprocessing does not increase the computational load significantly.

Thus the susceptibility $\gamma$ can be measured by comparing the evolution of two processes: one following the original process (2), and the other having a constant stimulus as in (3) with $\delta h_j(t) = \eta$ for all $j$. When the dynamics has reached the steady state, the measurement $\langle \tilde{w}_j - w_j \rangle$ yields the quantity $\eta \gamma$.

We illustrate the extension to two-layer networks by considering the committee machine, in which the error function takes the form $\epsilon(\sum_a f(x_a), y)$, where $a = 1, \cdots, n_h$ is the label of a hidden node, $x_a \equiv \vec{w}_a \cdot \vec{\xi}$ is the activation at the hidden node $a$, and $f$ represents the activation function. The generalization error is thus a function of the cavity activations of the hidden nodes, namely, $\hat{\epsilon}_g = \sum_\mu \epsilon(\sum_a f(h_a^\mu), y^\mu)/p$, where $h_a^\mu = \vec{w}_a^{\backslash\mu} \cdot \vec{\xi}^\mu$. When the inputs are independent and normalized, they are related to the generic activations by

$$h_a^\mu = x_a^\mu + \sum_b \gamma_{ab} \frac{\partial \epsilon(\sum_c f(x_c^\mu), y^\mu)}{\partial x_b^\mu}, \tag{9}$$

where $\gamma_{ab} = \lim_{t\to\infty} \int ds \, G_{ab}(t, s)$ is the susceptibility tensor. The Green's function $G_{ab}(t, s)$ represents the response of a weight feeding hidden node $a$ due to a stimulus applied at the gradient with respect to a weight feeding node $b$. It is obtained by monitoring $n_h + 1$ learning processes, one being original and each of the other $n_h$ processes having constant stimuli at the gradients with respect to one of the hidden nodes, viz.,

$$\frac{d\tilde{w}_{aj}^{(b)}(t)}{dt} = -\frac{1}{N}\frac{\partial E}{\partial \tilde{w}_{aj}^{(b)}} + \eta \delta_{ab}, \quad b = 1, \cdots, n_h. \tag{10}$$

When the dynamics has reached the steady state, the measurement $\langle \tilde{w}_{aj}^{(b)} - w_{aj} \rangle$ yields the quantity $\eta\gamma_{ab}$.

We will also compare the results with those obtained by extending the analysis of linear unlearning leave-one-out (LULOO) validation [6]. Consider the case that the regularization $R(\vec{w})$ takes the form of a weight decay term, $R(\vec{w}) = N \sum_{ab} \lambda_{ab} \vec{w}_a \cdot \vec{w}_b/2$. The cavity activations will be given by

$$h_a^\mu = x_a^\mu + \sum_b \left( \frac{\frac{1}{N}\sum_{jk}\xi_j^\mu(\Lambda+\mathbf{Q})_{aj,bk}^{-1}\xi_k^\mu}{1 - \frac{\epsilon_\mu''}{N}\sum_{cjdk}\xi_j^\mu f'(x_c^\mu)(\Lambda+\mathbf{Q})_{cj,dk}^{-1}f'(x_d^\mu)\xi_k^\mu} \right) \frac{\partial \epsilon(\sum_c f(x_c^\mu), y^\mu))}{\partial x_b^\mu}, \tag{11}$$

where $\epsilon_\mu''$ represents the second derivative of $\epsilon$ with respect to the student output for example $\mu$, and the matrix $\Lambda_{aj,bk} = \lambda_{ab}\delta_{jk}$ and $\mathbf{Q}$ is given by

$$Q_{aj,bk} = \frac{1}{N}\sum_\mu \xi_j^\mu f'(x_a^\mu)f'(x_b^\mu)\xi_k^\mu. \tag{12}$$

The LULOO result of (11) differs from the cavity result of (9) in that the susceptibility $\gamma_{ab}$ now depends on the example label $\mu$, and needs to be computed by inverting the matrix $\Lambda + \mathbf{Q}$. Note also that second derivatives of the error term have been neglected.

To verify the proposed method by simulations, we generate examples from a noisy teacher network which is a committee machine

$$y^\mu = \sum_{a=1}^{n_h} \mathrm{erf}\left(\frac{1}{\sqrt{2}}\vec{B}_a \cdot \vec{\xi}^\mu\right) + \sigma z_\mu. \tag{13}$$

Here $\vec{B}_a$ is the teacher vector at the hidden node $a$. $\sigma$ is the noise level. $\xi_j^\mu$ and $z_\mu$ are Gaussian variables with zero means and unit variances. Learning is done by the gradient descent of the energy function

$$E = \sum_\mu \frac{1}{2}\left[y^\mu - \sum_a \mathrm{erf}\left(\frac{1}{\sqrt{2}}\vec{w}_a \cdot \vec{\xi}^\mu\right)\right]^2 + \frac{1}{2}\lambda N \sum_a w_a^2 \tag{14}$$

and the weight decay parameter $\lambda$ is the hyperparameter to be optimized. The generalization error $\epsilon_g$ is given by

$$\epsilon_g = \left\langle \frac{1}{2} \left[ \sum_a \text{erf}\left(\frac{1}{\sqrt{2}}\vec{B}_a \cdot \vec{\xi}\right) + \sigma z - \sum_a \text{erf}\left(\frac{1}{\sqrt{2}}\vec{w}_a \cdot \vec{\xi}\right) \right]^2 \right\rangle, \quad (15)$$

where the averaging is performed over the distribution of input $\vec{\xi}$ and noise $z$. It can be computed analytically in terms of the inner products $Q_{ab} = \vec{w}_a \cdot \vec{w}_b$, $T_{ab} = \vec{B}_a \cdot \vec{B}_b$ and $R_{ab} = \vec{B}_a \cdot \vec{w}_b$ [10]. However, this target result is only known by the teacher, since $T_{ab}$ and $R_{ab}$ are not accessible by the student.

Figure 1 shows the simulation results of 4 randomly generated samples. Four results are compared: the target generalization error observed by the teacher, and those estimated by the cavity method, cross-validation and extended LULOO. It can be seen that the cavity method yields estimates of the optimal weight decay with comparable precision as the other methods.

For a more systematic comparison, we search for the optimal weight decay in 10 samples using golden section search [11] for the same parameters as in Fig. 1. Compared with the target results, the standard deviations of the estimated optimal weight decays are 0.3, 0.25 and 0.24 for the cavity method, sevenfold cross-validation and extended LULOO respectively. In another simulation of 80 samples of the single-layer perceptron, the estimated optimal weight decays have standard deviations of 1.2, 1.5 and 1.6 for the cavity method, tenfold cross-validation and extended LU-LOO respectively (the parameters in the simulations are $N = 500$, $p = 400$ and $\sigma$ ranging from 0.98 to 2.56).

To put these results in perspective, we mention that the computational resources needed by the cavity method is much less than the other estimations. For example, in the single-layer perceptrons, the CPU time needed to estimate the optimal weight decay using the golden section search by the teacher, the cavity method, tenfold cross-validation and extended LULOO are in the ratio of 1 : 1.5 : 3.0 : 4.6.

Before concluding this section, we mention that it is possible to derive an expression of the gradient $d\hat{\epsilon}_g/d\lambda$ of the estimated generalization error with respect to the weight decay. This provides us an even more powerful tool for hyperparameter estimation. In the case of the search for one hyperparameter, the gradient enables us to use the binary search for the zero of the gradient, which converges faster than the golden section search. In the single-layer experiment we mentioned, its precision is comparable to fivefold cross-validations, and its CPU time is only 4% more than the teacher's search. Details will be presented elsewhere. In the case of more than one hyperparameters, the gradient information will save us the need for an exhaustive search over a multidimensional hyperparameter space.

## 3   Dynamical Hyperparameter Estimation

The second example concerns the estimation of a dynamical hyperparameter, namely the optimal early stopping time, in cases where overtraining may plague the generalization ability at the steady state. In perceptrons, when the examples are noisy and the weight decay is weak, the generalization error decreases in the early stage of learning, reaches a minimum and then increases towards its asymptotic value [12, 9]. Since the early stopping point sets in before the system reaches the steady state, most analyses based on the equilibrium state are not applicable. Cross-validation stopping has been proposed as an empirical method to control overtraining [13]. Here we propose the cavity method as a convenient alternative.

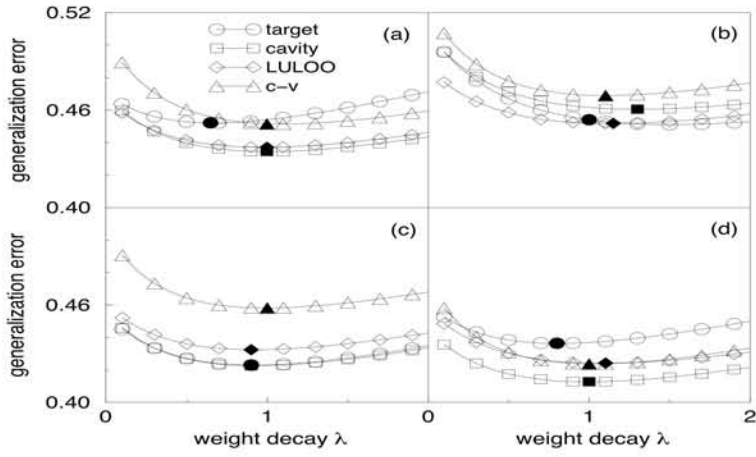

Figure 1: (a-d) The dependence of the generalization error of the multilayer percep-tron on the weight decay for $N = 200$, $p = 700$, $n_h = 3$, $\sigma = 0.8$ in 4 samples. The solid symbols locate the optimal weight decays estimated by the teacher (circle), the cavity method (square), extended LULOO (diamond) and sevenfold cross-validation (triangle).

In single-layer perceptrons, the cavity activations of the examples evolve according to (6), enabling us to estimate the dynamical evolution of the estimated general-ization error when learning proceeds. The remaining issue is the measurement of the time-dependent Green's function. We propose to introduce an initial homoge-neous stimulus, that is, $\delta h_j(t) = \eta\delta(t)$ for all $j$. Again, assuming normalized and independent inputs with $\langle \xi_j \rangle = 0$ and $\langle \xi_j \xi_k \rangle = \delta_{jk}$, we can see from (4) that the measurement $\langle \tilde{w}_j(t) - w_j(t) \rangle$ yields the quantity $\eta G(t, 0)$.

We will first consider systems that are time-translational invariant, i.e., $G(t, s) = G(t-s, 0)$. Such are the cases for Adaline learning and linear regression [9], where the cavity activation can be written as

$$h^\mu(t) = x^\mu(t) + \int ds\, G(t-s, 0)\frac{\partial \epsilon(x^\mu(s), y^\mu)}{\partial x^\mu(s)}. \qquad (16)$$

This allows us to estimate the generalization error $\hat{\epsilon}_g(t)$ via $\hat{\epsilon}_g(t) = \sum_\mu \epsilon(h^\mu(t), y^\mu)/p$, whose minimum in time determines the early stopping point.

To verify the proposed method in linear regression, we randomly generate exam-ples from a noisy teacher with $y^\mu = \vec{B} \cdot \vec{\xi}^\mu + \sigma z_\mu$. Here $\vec{B}$ is the teacher vec-tor with $B^2 = 1$. $\xi_j^\mu$ and $z_\mu$ are independently generated with zero means and unit variances. Learning is done by the gradient descent of the energy function $E(t) = \sum_\mu (y^\mu - \vec{w}(t) \cdot \vec{\xi}^\mu)^2/2$. The generalization error $\epsilon_g(t)$ is the error av-eraged over the distribution of input $\vec{\xi}$ and their corresponding output $y$, i.e., $\epsilon_g(t) = \langle (\vec{B} \cdot \vec{\xi} + \sigma z - \vec{w} \cdot \vec{\xi})^2/2 \rangle$. As far as the teacher is concerned, $\epsilon_g(t)$ can be computed as $\epsilon_g(t) = (1 - 2R(t) + Q(t) + \sigma^2)/2$. where $R(t) = \vec{w}(t) \cdot \vec{B}$ and $Q(t) = w(t)^2$.

Figure 2 shows the simulation results of 6 randomly generated samples. Three re-sults are compared: the teacher's estimate, the cavity method and cross-validation. Since LULOO is based on the equilibrium state, it cannot be used in the present

context. Again, we see that the cavity method yields estimates of the early stopping time with comparable precision as cross-validation. The ratio of the CPU time between the cavity method and fivefold cross-validation is $1 : 1.4$.

For nonlinear regression and multilayer networks, the Green's functions are not time-translational invariant. To estimate the Green's functions in this case, we have devised another scheme of stimuli. Preliminary results for the determination of the early stopping point are satisfactory and final results will be presented elsewhere.

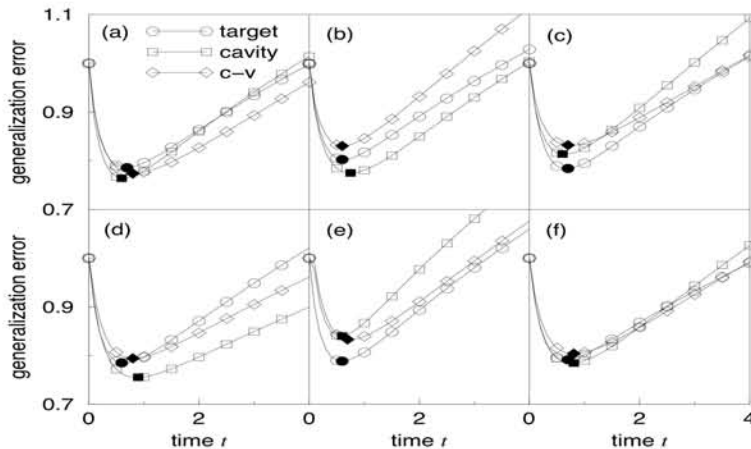

Figure 2: (a-f) The evolution of the generalization error of linear regression for $N = 500$, $p = 600$ and $\sigma = 1$. The solid symbols locate the early stopping points estimated by the teacher (circle), the cavity method (square) and fivefold cross-validation (diamond).

## 4 Conclusion

We have proposed a method for the fast estimation of hyperparameters in large networks, based on the linear response relation in the cavity method, combined with an empirical method of measuring the Green's function. Its efficiency depends on the independent and identical distribution of the inputs, greatly reducing the number of networks to be monitored. It does not require the validation process or the inversion of matrices of macroscopic size, and hence its speed compares favorably with cross-validation and other perturbative approaches such as extended LULOO. For multilayer networks, we will explore further speedup of the Green's function measurement by multiplexing the stimuli to the different hidden units into a single network, to be compared with a reference network. We will also extend the technique to other benchmark data to study its applicability.

Our initial success indicates that it is possible to generalize the method to more complicated systems in the future. The concept of Green's functions is very general, and its measurement by comparing the states of a stimulated system with a reference one can be adopted to general cases with suitable adaptation. Recently, much attention is paid to the issue of model selection in support vector machines [3, 4, 5]. It would be interesting to consider how the proposed method can contribute to these cases.

## Acknowledgements

We thank C. Campbell for interesting discussions and H. Nishimori for encouragement. This work was supported by the grant HKUST6157/99P from the Research Grant Council of Hong Kong.

## References

[1] C. M. Bishop, *Neural Networks for Pattern Recognition*, Clarendon Press, Oxford (1995).

[2] G. B. Orr and K.-R. Müller, eds., *Neural Networks: Tricks of the Trade*, Springer, Berlin (1998).

[3] O. Chapelle and V. N. Vapnik, *Advances in Neural Information Processing Systems* **12**, S. A. Solla, T. K. Leen and K.-R. Müller, eds., MIT Press, Cambridge, 230 (2000).

[4] S. S. Keerthi, Technical Report CD-01-02, *http://guppy.mpe.nus.edu.sg/ mpessk/nparm.html* (2001).

[5] M. Opper and O. Winther, *Advances in Large Margin Classifiers*, A. J. Smola, P. Bartlett, B. Schölkopf and D. Schuurmans, eds., MIT Press, Cambridge, 43 (1999).

[6] J. Larsen and L. K. Hansen, *Advances in Computational Mathematics* **5**, 269 (1996).

[7] M. Opper and O. Winther, *Phys. Rev. Lett.* **76**, 1964 (1996).

[8] A. L. Fetter and J. D. Walecka, *Quantum Theory of Many-Particle Systems*, McGraw-Hill, New York (1971).

[9] K. Y. M. Wong, S. Li and Y. W. Tong, *Phys. Rev. E* **62**, 4036 (2000).

[10] D. Saad and S. A. Solla, *Phys. Rev. Lett.* **74**, 4337 (1995).

[11] W. H. Press, B. P. Flannery, S. A. Teukolsky and W. T. Vetterling, *Numerical Recipes in C: The Art of Scientific Computing*, Cambridge University Press, Cambridge (1990).

[12] A. Krogh and J. A. Hertz, *J. Phys. A* **25**, 1135 (1992).

[13] S. Amari, N. Murata, K.-R. Müller, M. Finke and H. H. Yang, *IEEE Trans. on Neural Networks* **8**, 985 (1997).
